# Dual Inhibitory Mechanisms for Definition of Receptive Field Characteristics in Cat Striate Cortex

**A. B. Bonds**
Dept. of Electrical Engineering
Vanderbilt University
Nashville, TN 37235

## Abstract

In single cells of the cat striate cortex, lateral inhibition across orientation and/or spatial frequency is found to enhance pre-existing biases. A contrast-dependent but spatially non-selective inhibitory component is also found. Stimulation with ascending and descending contrasts reveals the latter as a response hysteresis that is sensitive, powerful and rapid, suggesting that it is active in day-to-day vision. Both forms of inhibition are not recurrent but are rather network properties. These findings suggest two fundamental inhibitory mechanisms: a global mechanism that limits dynamic range and creates spatial selectivity through thresholding and a local mechanism that specifically refines spatial filter properties. Analysis of burst patterns in spike trains demonstrates that these two mechanisms have unique physiological origins.

## 1 INFORMATION PROCESSING IN STRIATE CORTICAL CELLS

The most popular current model of single cells in the striate cortex casts them in terms of spatial and temporal filters. The input to visual cortical cells from lower visual areas, primarily the LGN, is fairly broadband (e.g., Soodak, Shapley & Kaplan, 1987; Maffei & Fiorentini, 1973). Cortical cells perform significant bandwidth restrictions on this information in at least three domains: orientation, spatial frequency and temporal frequency. The most interesting quality of these cells is

therefore what they reject from the broadband input signal, rather than what they pass, since the mere passage of the signal adds no information. Visual cortical cells also show contrast-transfer, or amplitude-dependent, nonlinearities which are not seen at lower levels in the visual pathway. The primary focus of our lab is study of the cortical mechanisms that support both the band limitations and nonlinearities that are imposed on the relatively unsullied signals incoming from the LGN. All of our work is done on the cat.

## 2    THE ROLE OF INHIBITION IN ORIENTATION SELECTIVITY

Orientation selectivity is one of the most dramatic demonstrations of the filtering ability of cortical cells. Cells in the LGN are only mildly biased for stimulus orientation, but cells in cortex are completely unresponsive to orthogonal stimuli and have tuning bandwidths that average only about 40-50° (e.g., Rose & Blakemore, 1974). How this happens remains controversial, but there is general consensus that inhibition helps to define orientation selectivity although the schemes vary. The concept of *cross-orientation inhibition* suggests that the inhibition is itself orientation selective and tuned in a complimentary way to the excitatory tuning of the cell, being smallest at the optimal orientation and greatest at the orthogonal orientation. More recent results, including those from our own lab, suggests that this is not the case.

We studied the orientation dependence of inhibition by presenting two superimposed gratings, a *base* grating at the optimal orientation to provide a steady level of background response activity, and a *mask* grating of varying orientation which yielded either excitation or inhibition that could supplement or suppress the *base*-generated response. There is some confusion when both base and mask generate excitation. In order to separate the response components from each of these stimuli, the two gratings were drifted at differing temporal frequencies. At least in simple cells, the individual contributions to the response from each grating could then be resolved by performing Fourier analysis on the response histograms.

Experiments were done on 52 cells, of which about 2/3 showed organized suppression from the *mask* grating (Bonds, 1989). Fig. 1 shows that while the mask-generated response suppression is somewhat orientation selective, it is by and large much flatter than would be required to account for the tuning of the cell. There is thus *some* orientation dependence of inhibition, but not specifically at the orthogonal orientation as might be expected. Instead, the predominant component of the suppression is constant with mask orientation, or **global**. This suggests that virtually any stimulus can result in inhibition, whether or not the recorded cell actually "sees" it. What orientation-dependent component of inhibition that might appear is expressed in suppressive side-bands near the limits of the excitatory tuning function, which have the effect of enhancing any pre-existing orientation bias.

Thus the concept of cross-orientation inhibition is not particularly correct, since the inhibition is found not just at the "cross" orientation but rather at all orientations. Even without orientation-selective inhibition, a scheme for establishment of true orientation selectivity from orientation-biased LGN input can be derived

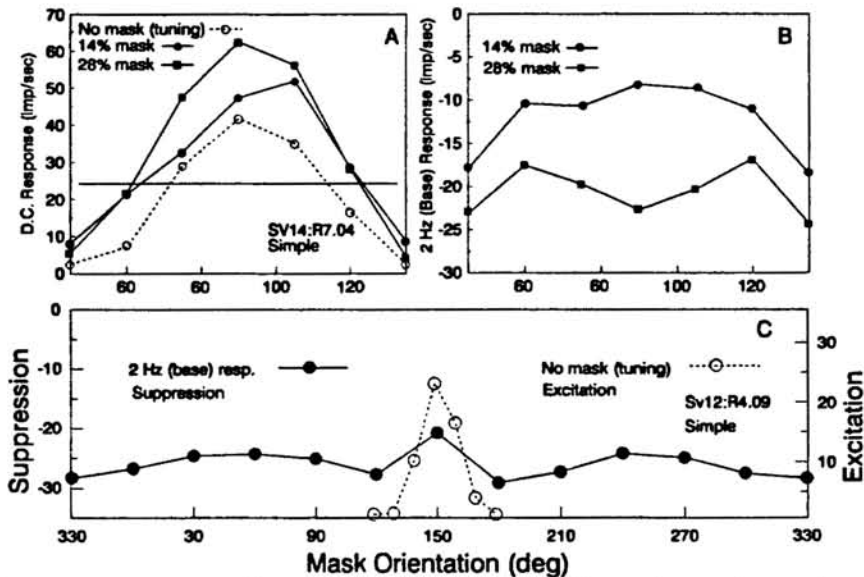

Figure 1: Response suppression by mask gratings of varying orientation. A. Impact of masks of 2 different contrasts on 2 Hz (base-generated) response, expressed by decrease (negative imp/sec) from response level arising from base stimulus alone. B. Similar example for mask orientations spanning a full 360°.

by assuming that the nonselective inhibition is graded and contrast-dependent and that it acts as a thresholding device (Bonds, 1989).

# 3   THE ROLE OF INHIBITION IN SPATIAL FREQUENCY SELECTIVITY

While most retinal and LGN cells are broadly tuned and predominantly low-pass, cortical cells generally have spatial frequency bandpasses of about 1.5-2 octaves (e.g., Maffei & Fiorentini, 1973). We have examined the influence of inhibition on spatial frequency selectivity using the same strategy as the previous experiment (Bauman & Bonds, 1991). A *base* grating, at the optimal orientation and spatial frequency, drove the cell, and a superimposed *mask* grating, at the optimal orientation but at different spatial and temporal frequencies, provided response facilitation or suppression.

We defined three broad categories of spatial frequency tuning functions: Low pass, with no discernible low-frequency fall-off, band-pass, with a peak between 0.4 and 0.9 c/deg, and high pass, with a peak above 1 c/deg. About 75% of the cells showed response suppression organized with respect to the spatial frequency of mask gratings. For example, Fig. 2A shows a low-pass cells with high-frequency suppression and Fig. 2B shows a band-pass cell with mixed suppression, flanking the tuning curve at both low and high frequencies. In each case response suppression was graded with mask contrast and some suppression was found even at the optimal spatial frequency. Some cells showed no suppression, indicating that the suppression was not merely a stimulus artifact. In all but 2 of 42 cases, the suppression was appropriate to the enhancement of the tuning function (e.g., low-pass cells had high-frequency response suppression), suggesting that the design of the system is more

than coincidental. No similar spatial-frequency-dependent suppression was found in LGN cells.

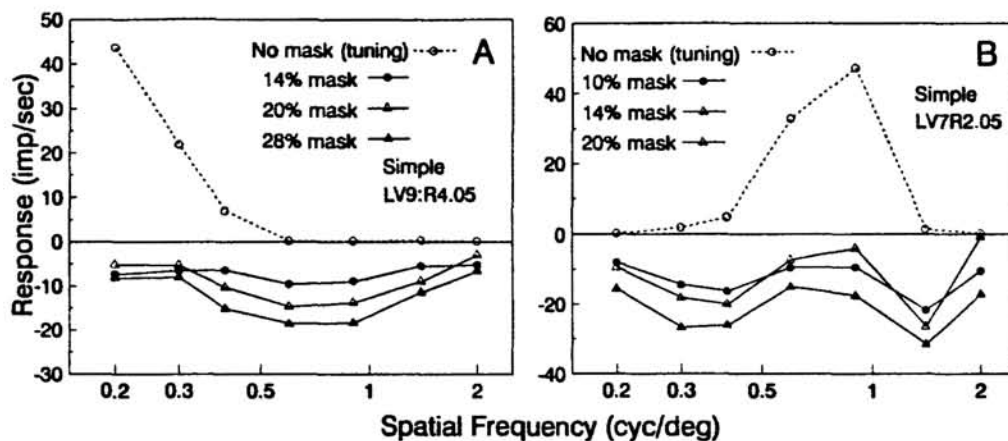

Figure 2: Examples of spatial frequency-dependent response suppression. Upper broken lines show excitatory tuning functions and solid lines below zero indicate response reduction at three different contrasts. A. Low-pass cell with high frequency inhibition. B. Band-pass cell with mixed (low and high frequency) inhibition. Note suppression at optimal spatial frequency in both cases.

## 4    NON-STATIONARITY OF CONTRAST TRANSFER PROPERTIES

The two experiments described above demonstrate the existence of intrinsic cortical mechanisms that refine the spatial filter properties of the cells. They also reveal a *global* form of inhibition that is spatially non-specific. Since it is found even with spatially optimal stimuli, it can influence the form of the cortical contrast-response function (usually measured with optimal stimuli). This function is essentially logarithmic, with saturation or even super-saturation at higher contrasts (e.g., Albrecht & Hamilton, 1982), as opposed to the more linear response behavior seen in cells earlier in the visual pathway. Cortical cells also show some degree of contrast adaptation; when exposed to high mean contrasts for long periods of time, the response vs contrast curves move rightward (e.g., Ohzawa, Sclar & Freeman, 1985). We addressed the question of whether contrast-response nonlinearity and adaptation might be causally related.

In order to compensate for "intrinsic response variability" in visual cortical cells, experimental stimulation has historically involved presentation of randomized sequences of pattern parameters, the so-called multiple histogram technique (Henry, Bishop, Tupper & Dreher, 1973). Scrambling presentation order distributes time-dependent response variability across all stimulus conditions, but this procedure can be self-defeating by masking any stimulus-dependent response variation. We therefore presented cortical cells with ordered sequences of contrasts, first ascending then descending in a stepwise manner (Bonds, 1991). This revealed a clear and powerful response hysteresis. Fig. 3A shows a solid line representing the contrast-response

function measured in the usual way, with randomized parameter presentation, over-laid on an envelope outlining responses to sequentially increasing or decreasing 3-sec contrast epochs; one sequential presentation set required 54 secs. Across 36 cells measured in this same way, the average response hysteresis corresponded to 0.36 log units of contrast. Some hysteresis was found in every cortical cell and in no LGN cells, so this phenomenon is intrinsically cortical.

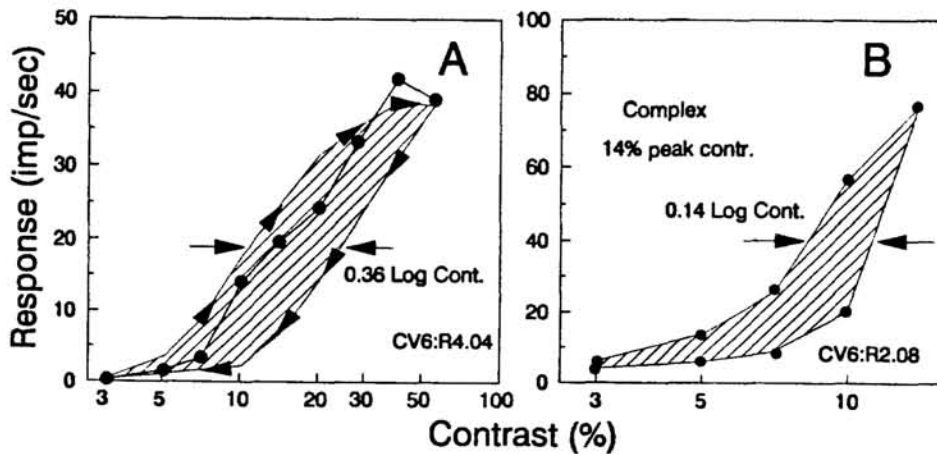

Figure 3: Dynamic response hysteresis. **A.** A response function measured in the usual way, with randomized stimulus sequences (filled circles) is overlaid on the function resulting from stimulation with sequential ascending (upper level) and descending (lower level) contrasts. Each contrast was presented for 3 seconds. **B.** Hysteresis resulting from peak contrast of 14%; 3 secs per datum.

Hysteresis demonstrates a clear dependence of response amplitude on the history of stimulation: at a given contrast, the amplitude is always less if a higher contrast was shown first. This is one manifestation of cortical contrast adaptation, which is well-known. However, adaptation is usually measured after long periods of stim-ulation with high contrasts, and may not be relevant to normal behavioral vision. Fig. 3B shows hysteresis at a modest response level and low peak contrast (14%), suggesting that it can serve a major function in day-to-day visual processing. The speed of hysteresis also addresses this issue, but it is not so easily measured. Some response histogram waveforms show consistent amplitude loss over a few seconds of stimulation (see also Albrecht, Farrar & Hamilton, 1984), but other histograms can be flat or even show a slight rise over time despite clear contrast adaptation (Bonds, 1991). This suggests the possibility that, in the classical pattern of any well-designed automatic gain control, gain reduction takes place quite rapidly, but its effects linger for some time.

The speed of reaction of gain change is illustrated in the experiment of Fig. 4. A "pedestal" grating of 14% contrast is introduced. After 500 msec, a contrast increment of 14% is added to the pedestal for a variable length of time. The response during the first and last 500 msec of the pedestal presentation is counted and the ratio is taken. In the absence of the increment, this ratio is about 0.8, reflecting the adaptive nature of the pedestal itself. For an increment of even 50 msec duration, this ratio is reduced, and it is reduced monotonically–by up to half the control

level–for increments lasting less than a second. The gain control mechanism is thus both sensitive and rapid.

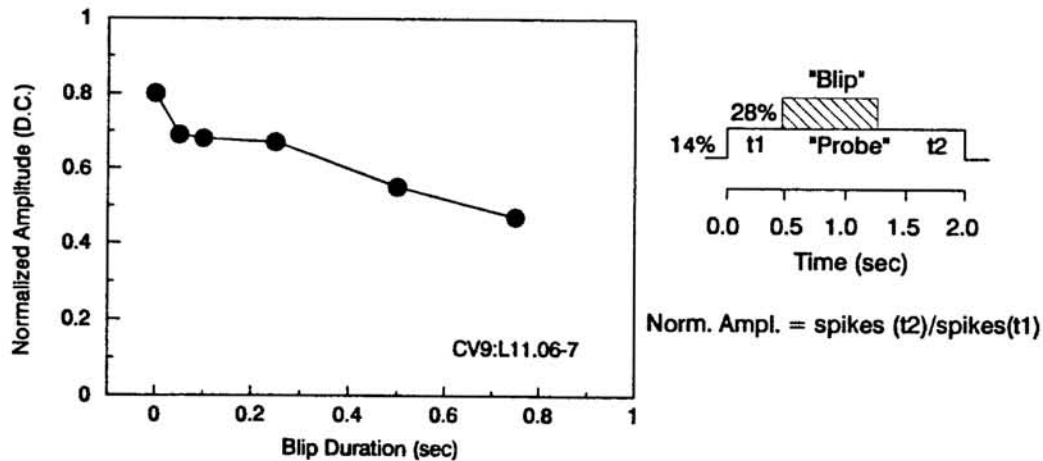

Figure 4: Speed of gain reduction. The ratio of spikes generated during the last and first 500 msec of a 2 sec pedestal presentation can be modified by a brief contrast increment (see text).

# 5    PHYSIOLOGICAL INDEPENDENCE OF TWO INHIBITORY MECHANISMS

The experimental observations presented above support two basic phenomena: spatially-dependent and spatially-independent inhibition. The question remains whether these two types of inhibition are fundamentally different, or if they stem from the same physiological mechanisms. This question can be addressed by examining the structure of a serial spike train generated by a cortical cell. In general, rather than being distributed continuously, cortical spikes are grouped into discrete packets, or bursts, with some intervening isolated spikes. The burst structure can be fundamentally characterized by two parameters: the burst frequency (*bursts per second*, or BPS) and the burst duration (*spikes per burst*, or SPB).

We have analyzed cortical spike trains for these properties by using an adaptive algorithm to define burst groupings; as a rule of thumb, spike intervals of 8 msec or less were considered to belong to bursts. Both burst frequency (BPS) and structure (SPB) depend strongly on mean firing rate, but once firing rate is corrected for, two basic patterns emerge. Consider two experiments, both yielding firing rate variation about a similar range. In one experiment, firing rate is varied by varying stimulus contrast, while in the other, firing rate is varied by varying stimulus orientation. Burst frequency (BPS) depends only on spike rate, regardless of the type of experiment. In Fig. 5A, no systematic difference is seen between the experiments in which contrast (filled circles) and orientation (open squares) are varied. To quantify the difference between the curves, polynomials were fit to each and the quantity gamma, defined by the (shaded) area bounded by the two polynomials, was calculated; here, it equalled about 0.03.

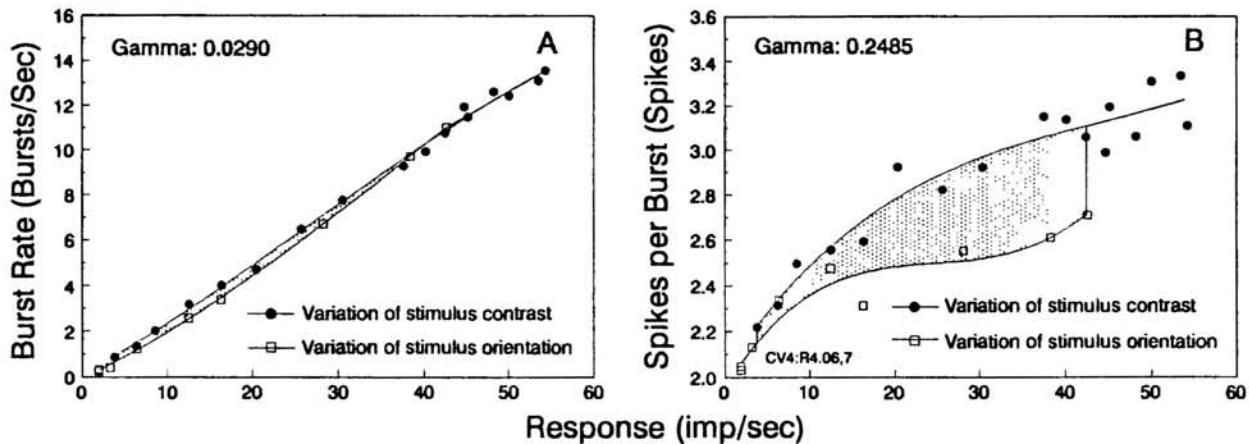

Figure 5: A. Comparison of burst frequency (bursts per second) as function of firing rate resulting from presentations of varying contrast (filled circles) and varying orientation (open squares). B. Comparison of burst length (spikes per burst) under similar conditions. Note that at a given firing rate, burst length is always shorter for experiment parametric on orientation. Shaded area (*gamma*) is quantitative indicator of difference between two curves.

Fig. 5B shows that at similar firing rates, burst length (SPB) is markedly shorter when firing rate is controlled by varying orientation (open squares) rather than contrast (filled circles). In this pair of curves, the gamma (of about 0.25) is nearly ten times that found in the upper curve. This is a clear violation of univariance, since at a given spike rate (output level), the structure of the spike train differs depending on the type of stimulation. Analysis of cortical response merely on the basis of overall firing rate thus does not give the signalling mechanisms the respect they are properly due. This result also implies that the strength of signalling between nerve cells can dynamically vary independent of firing rate. Because of post-synaptic temporal integration, bursts of spikes with short interspike intervals will be much more effective in generating depolarization than spikes at longer intervals. Thus, at a given average firing rate, a cell that generates longer bursts will have more influence on a target cell than a cell that distributes its spikes in shorter bursts, all other factors being equal.

This phenomenon was consistent across a population of 59 cells. Gamma, which reflects the degree of difference between curves measured by variation of contrast and by variation of orientation, averaged zero for curves based on number of bursts (BPS). For both simple and complex cells, gamma for burst duration (SPB) averaged 0.15.

At face value, these results simply mean that when lower spike rates are achieved by use of non-optimal orientations, they result from shorter bursts than when lower spike rates result from reduction of contrast (with the spatial configuration remaining optimal). This means that non-optimal orientations and, from some preliminary results, non-optimal spatial frequencies, result in inhibition that acts specifically to shorten bursts, whereas contrast manipulations for the most part act to modulate both the number and length of bursts.

These results suggest strongly that there are at least two distinct forms of cortical inhibition, with unique physiological bases differentiated by the burst organization in cortical spike trains. Recent results from our laboratory (Bonds, *Unpub. Obs.*) confirm that burst length modulation, which seems to reflect inhibition that depends on the spatial characteristics of the stimulus, is strongly mediated by GABA. Microiontophoretic injection of GABA shortens burst length and injection of bicuculline, a GABA blocker, lengthens bursts. This is wholly consistent with the hypothesis that GABA is central to definition of spatial qualities of the cortical receptive field, and suggests that one can indirectly observe GABA-mediated inhibition by spike train analysis.

## Acknowledgements

This work was done in collaboration with Ed DeBruyn, Lisa Bauman and Brian DeBusk. Supported by NIH (RO1-EY03778-09).

## References

D. G. Albrecht & D. B. Hamilton. (1982) Striate cortex of monkey and cat: contrast response functions. *Journal of Neurophysiology* **48**, 217-237.

D. G. Albrecht, S. B. Farrar & D. B. Hamilton. (1984) Spatial contrast adaptation characteristics of neurones recorded in the cat's visual cortex. *Journal of Physiology* **347**, 713-739.

A. B. Bonds. (1989) The role of inhibition in the specification of orientation selectivity of cells of the cat striate cortex. *Visual Neuroscience* **2**, 41-55.

A. B. Bonds. (1991) Temporal dynamics of contrast gain control in single cells of the cat striate cortex. *Visual Neuroscience* **6**, 239-255.

L. A. Bauman & A. B. Bonds. (1991) Inhibitory refinement of spatial frequency selectivity in single cells of the cat striate cortex. *Vision Research* **31**, 933-944.

G. Henry, P. O. Bishop, R. M. Tupper & B. Dreher. (1973) Orientation specificity of cells in cat striate cortex. *Vision Research* **13**, 1771-1779.

L. Maffei & A. Fiorentini. (1973) The visual cortex as a spatial frequency analyzer. *Vision Research* **13**, 1255-1267.

I. Ohzawa, G. Sclar & R. D. Freeman. (1985) Contrast gain control in the cat's visual system. *Journal of Neurophysiology* **54**, 651-667.

D. Rose & C. B. Blakemore. (1974) An analysis of orientation selectivity in the cat's visual cortex. *Experimental Brain Research* **20**, 1-17.

R. E. Soodak, R. M. Shapley & E. Kaplan. (1987) Linear mechanism of orientation tuning in the retina and lateral geniculate of the cat. *Journal of Neurophysiology* **58**, 267-275.
